# Approximate Learning of Dynamic Models

**Xavier Boyen**
Computer Science Dept. 1A
Stanford, CA 94305-9010
*xb@cs.stanford.edu*

**Daphne Koller**
Computer Science Dept. 1A
Stanford, CA 94305-9010
*koller@cs.stanford.edu*

## Abstract

Inference is a key component in learning probabilistic models from partially observable data. When learning temporal models, each of the many inference phases requires a traversal over an entire long data sequence; furthermore, the data structures manipulated are exponentially large, making this process computationally expensive. In [2], we describe an approximate inference algorithm for monitoring stochastic processes, and prove bounds on its approximation error. In this paper, we apply this algorithm as an approximate forward propagation step in an EM algorithm for learning temporal Bayesian networks. We provide a related approximation for the backward step, and prove error bounds for the combined algorithm. We show empirically that, for a real-life domain, EM using our inference algorithm is much faster than EM using exact inference, with almost no degradation in quality of the learned model. We extend our analysis to the online learning task, showing a bound on the error resulting from restricting attention to a small window of observations. We present an online EM learning algorithm for dynamic systems, and show that it learns much faster than standard offline EM.

## 1 Introduction

In many real-life situations, we are faced with the task of inducing the dynamics of a complex stochastic process from limited observations about its state over time. Until now, *hidden Markov models (HMMs)* [12] have played the largest role as a representation for learning models of stochastic processes. Recently, however, there has been increasing use of more structured models of stochastic processes, such as *factorial HMMs* [8] or *dynamic Bayesian networks (DBNs)* [4]. Such structured decomposed representations allow complex processes over a large number of states to be encoded using a much smaller number of parameters, thereby allowing better generalization from limited data [8, 7, 13]. Furthermore, the natural structure of such processes makes it easier for a human expert to incorporate prior knowledge about the domain structure into the model, thereby improving its inductive bias.

Both parameter and structure learning algorithms for dynamic models [12, 7] use probabilistic inference as a crucial component. An inference routine is called multiple times in order to "fill in" missing data with its expected value according to the current hypothesis; the resulting *expected sufficient statistics* are then used to construct a new hypothesis. The inference step is used many times, each of which iterates over the entire sequence. This behavior is problematic in two important respects. First, in many settings, we may not have access to the entire sequence in advance. Second, the various structured representations of stochastic processes do not admit an effective inference procedure. The messages propagated by exact inference algorithms include an entry for each possible state of the system; the number of states is exponential in the size of our model, rendering this type of computation infeasible in all but the smallest of problems. In this paper, we describe and analyze an approach that helps us address both of these problems.

In [2], we proposed a new approach to approximate inference in stochastic processes, where *approximate* distributions that admit compact representation are maintained and propagated. Our approach can achieve exponential savings over exact inference for DBNs. We showed empirically that, for a practical DBN [6], our approach results in a factor 15–20 reduction in running time at only a small cost in accuracy. We also proved that the accumulated error arising from the repeated approximations remains bounded indefinitely over time. This result relied on an analysis showing that transition through a stochastic process is a contraction for relative entropy (KL-divergence) [3].

Here, we apply this approach to the parameter learning task. This application is not completely straightforward, since our algorithm of [2] and the associated analysis only applied to the forward propagation of messages, whereas the inference used in learning algorithms require propagation of information from the entire sequence. In this paper, we provide an analysis of the error accumulated by an approximate inference process in the backward propagation phase of inference. This analysis is quite different from the contraction analysis for the forward phase. We combine these two results to prove bounds on the error of the expected sufficient statistics relayed to the learning algorithm at each stage. We then present empirical results for a practical DBN, illustrating the performance of this approximate learning algorithm. We show that speedups of 15–20 can be obtained easily, with no discernable loss in the quality of the learned hypothesis.

Our theoretical analysis also suggests a way of dealing with the problematic need to reason about the entire sequence of temporal observations at once. Our contraction results show that it is legitimate to ignore observations that are very far in the future. Thus, we can compute a very accurate approximation to the backward message by considering only a small window of observations in the future. This idea leads to an efficient online learning algorithm. We show that it converges to a good hypothesis much faster than the standard offline EM algorithm, even in settings favorable to the latter.

## 2 Preliminaries

A model for a dynamic system is specified as a tuple $\langle \mathcal{B}, \Theta \rangle$ where $\mathcal{B}$ represents the qualitative structure of the model, and $\Theta$ the appropriate parameterization. In a DBN, the instantaneous state of a process is specified in terms of a set of variables $X_1, ..., X_n$. Here, $\mathcal{B}$ encodes a network fragment which specifies, for each time $t$ variable $X_k^{(t)}$, the set of parents $Parents(X_k^{(t)})$; an example fragment is shown in Figure 1(a). The parameters $\Theta$ define for each $X_k^{(t)}$ a conditional probability table $P[X_k^{(t)} \mid Parents(X_k^{(t)})]$. For simplicity, we assume that the variables are partitioned into *state* variables, which are never observed, and *observation* variables, which are always observed. We also assume that the observation variables at time $t$ depend only on state variables at time $t$. We use $\mathcal{T}$ to denote the transition matrix over the state variables in the stochastic process; i.e., $\mathcal{T}_{i,j}$ is the transition probability

from state $s_i$ to state $s_j$. Note that this concept is well-defined even for a DBN, although in that case, the matrix is represented implicitly via the other parameters. We use $\mathcal{O}$ to denote the observation matrix; i.e., $\mathcal{O}_{i,j}$ is the probability of observing response $r_j$ in state $s_i$.

Our goal is to learn the model for stochastic process from partially observable data. To simplify our discussion, we focus on the problem of learning parameters for a known structure using the *EM (Expectation Maximization)* algorithm [5]; most of our discussion applies equally to other contexts (e.g., [7]). EM is an iterative procedure that searches over the space of parameter vectors for one which is a local maximum of the likelihood function— the probability of the observed data $D$ given $\Theta$. We describe the EM algorithm for the task of learning HMMs; the extension to DBNs is straightforward. The EM algorithm starts with some initial (often random) parameter vector $\tilde{\Theta}$, which specifies a current estimate of the transition and observation matrices of the process $\tilde{\mathcal{T}}$ and $\tilde{\mathcal{O}}$. The EM algorithm computes the *expected sufficient statistics (ESS)* for $D$, using $\tilde{\mathcal{T}}$ and $\tilde{\mathcal{O}}$ to compute the expectation. In the case of HMMs, the ESS are an average, over $t$, of the joint distributions $\tilde{\psi}^{(t)}$ over the variables at time $t-1$ and the variables at time $t$. A new parameter vector $\tilde{\Theta}'$ can then be computed from the ESS by a simple maximum likelihood step. These two steps are iterated until an appropriate stopping condition is met.

The $\tilde{\psi}^{(t)}$ for the entire sequence can be computed by a simple forward-backward algorithm. Let $r^{(t)}$ be the response observed at time $t$, and let $\mathcal{O}_{r^{(t)}}$ be its likelihood vector ($\mathcal{O}_{r_j}(i) \triangleq \mathcal{O}_{i,j}$). The forward messages $\alpha^{(t)}$ are propagated as follows: $\alpha^{(t)} \propto (\alpha^{(t-1)} \cdot \mathcal{T}) \times \mathcal{O}_{r^{(t)}}$, where $\times$ is the outer product. The backward messages $\beta^{(t)}$ are propagated as $\beta^{(t)} \propto (\mathcal{T} \cdot (\beta^{(t+1)} \times \mathcal{O}_{r^{(t+1)}})')'$. The estimated belief at time $t$ is now simply $\alpha^{(t)} \times \beta^{(t)}$ (suitably renormalized); similarly, the joint belief $\tilde{\psi}^{(t)}$ is proportional to $(\alpha^{(t-1)} \times \beta^{(t)} \times \mathcal{T} \times \mathcal{O}_{r^{(t)}})$.

This message passing algorithm has an obvious extension to DBNs. Unfortunately, it is feasible only for very small DBNs. Essentially, the messages passed in this algorithm have an entry for every possible state at time $t$; in a DBN, the number of states is exponential in the number of state variables, rendering such an explicit representation infeasible in most cases. Furthermore even highly structured processes do not admit a more compact representation of these messages [8, 2].

## 3  Belief state approximation

In [2], we described a new approach to approximate inference in dynamic systems, which avoids the problem of explicitly maintaining distributions over large spaces. We maintain our *belief state* (distribution over the current state) using some computationally tractable representation of a distribution. We propagate the time $t$ approximate belief state through the transition model and condition it on our evidence at time $t+1$. We then approximate the resulting time $t+1$ distribution using one that admits a compact representation, allowing the algorithm to continue. We also showed that the errors arising from the repeated approximation do not accumulate unboundedly, as the stochasticity of the process attenuates their effect.

In particular, for DBNs we considered belief state approximations where certain subsets of less correlated variables are grouped into distinct clusters which are approximated as being independent. In this case, the approximation at each step consists of a simple projection onto the relevant marginals, which are used as a factored representation of the time $t+1$ approximate belief state. This algorithm can be implemented efficiently using the clique tree algorithm [10]. To compute $\tilde{\alpha}^{(t+1)}$ from $\tilde{\alpha}^{(t)}$, we generate a clique tree over these two time slices of the DBN, ensuring that both the time $t$ and time $t+1$ clusters appear as a subset of some clique. We then incorporate $\tilde{\alpha}^{(t)}$ into the time $t$ cliques; $\tilde{\alpha}^{(t+1)}$ is obtained

by calibrating the tree (doing inference) and reading off the relevant marginals from the tree ($\tilde{\alpha}^{(t+1)}$ is implicitly defined as their product).

These results are directly applicable to the learning task, as the belief state is the forward message in the forward-backward algorithm. Thus, we can apply this approach to the forward step, with the guarantee that the approximation will not lead to a big difference in the ESS. However, this technique does not resolve our computational problems, as the backward propagation phase is as expensive as the forward phase. We can apply the same idea to the backward propagation, i.e., we maintain and propagate a compactly represented approximate backward message $\tilde{\beta}^{(t)}$. The implementation of this idea is a simple extension of our algorithm for forward messages. To compute $\tilde{\beta}^{(t)}$ from $\tilde{\beta}^{(t+1)}$, we simply incorporate $\tilde{\beta}^{(t+1)}$ into our clique tree over these two time slices, then read off the relevant marginals for computing $\tilde{\beta}^{(t)}$.

However, extending the analysis is not as straightforward. It is not completely straightforward to apply the techniques of [2] to get relative error bounds for the backward message. Furthermore, even if we have bounds on relative entropy error of both the forward and backward messages, bounds for the error of the $\psi^{(t)}$ do not follow. The solution turns out to use an alternative notion of distance, which combines additively under Bayesian updating, albeit at the cost of weaker contraction rates.

**Definition 1** Let $\rho$ and $\tilde{\rho}$ be two positive vectors of same dimension. Their *projective distance* is defined as $D_{\text{Proj}}[\rho, \tilde{\rho}] \triangleq \max_{i,i'} \ln[(\rho_i \cdot \tilde{\rho}_{i'})/(\rho_{i'} \cdot \tilde{\rho}_i)]$.

We note that the projective distance is a (weak) upper bound on the relative entropy.

**Lemma 1** $D_{\text{Proj}}[\psi^{(t)}, \tilde{\psi}^{(t)}] \leq D_{\text{Proj}}[\alpha^{(t-1)}, \tilde{\alpha}^{(t-1)}] + D_{\text{Proj}}[\beta^{(t)}, \tilde{\beta}^{(t)}]$.

Based on the results of [1], we show that projective distance contracts when messages are propagated through the stochastic transition matrix, in either direction. Of course, the rate of contraction depends on ergodicity properties of the matrix:

**Lemma 2** Let $k = \min_{\{i,j,i',j': \mathcal{T}_{i,j} \cdot \mathcal{T}_{i',j'} \neq 0\}} \sqrt{(\mathcal{T}_{i,j'} \cdot \mathcal{T}_{i',j})/(\mathcal{T}_{i,j} \cdot \mathcal{T}_{i',j'})}$, and define $\kappa_{\mathcal{T}} \triangleq 2 \cdot k/(1+k)$. Then $D_{\text{Proj}}[\alpha^{(t)}, \tilde{\alpha}^{(t)}] \leq (1 - \kappa_{\mathcal{T}}) \cdot D_{\text{Proj}}[\alpha^{(t-1)}, \tilde{\alpha}^{(t-1)}]$, and $D_{\text{Proj}}[\beta^{(t)}, \tilde{\beta}^{(t)}] \leq (1 - \kappa_{\mathcal{T}}) \cdot D_{\text{Proj}}[\beta^{(t+1)}, \tilde{\beta}^{(t+1)}]$.

We can now show that, if our approximations do not introduce too large an error, then the expected sufficient statistics will remain close to their correct value.

**Theorem 3** *Let $S$ be the ESS computed via exact inference, and let $\tilde{S}$ be its approximation. If the forward (backward) approximation step is guaranteed to introduce at most $\varepsilon$ ($\delta$) projective error, then $D_{\text{Proj}}[S, \tilde{S}] \leq (\varepsilon + \delta)/\kappa_{\mathcal{T}}$. Therefore $D_{\text{KL}}[S \| \tilde{S}] \leq (\varepsilon + \delta)/\kappa_{\mathcal{T}}$.*

Note that even small fluctuations in the sufficient statistics can cause the EM algorithm to reach a different local maximum. Thus, we cannot analytically compare the quality of the resulting algorithms. However, as our experimental results show, there is no divergence between exact EM and aproximate EM in practice.

We tested our algorithms on the task of learning the parameters for the BAT network shown in Figure 1(a), used for traffic monitoring [6]. The training set was a fixed sequence of 1000 slices, generated from the correct network distribution. Our test metric was the average log-likelihood (per slice) of a fixed test sequence of 50 slices. All experiments were conducted using three different random starting points for the parameters (the same

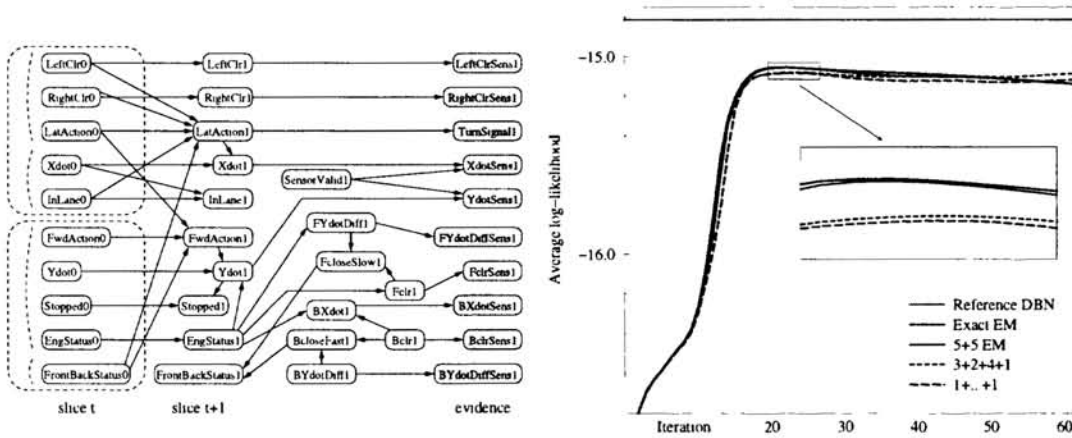

Figure 1: (a) The BAT DBN. (b) Structural approximations for batch EM.

in all the experiments). We ran EM with different types of structural approximations, and evaluated the quality of the model after each iteration of the algorithm. We used four different structural approximations: (i) exact propagation; (ii) a 5+5 clustering of the ten state variables; (iii) a 3+2+4+1 clustering; (iv) each variable in a separate cluster. The results for one random starting point are shown on Figure 1(b). As we can see, the impact of (even severe) structural approximation on learning accuracy is negligible. In all of the runs, the approximate algorithm tracked the exact one very closely, and the largest difference in the peak log-likelihood was at most 0.04. This phenomenon is rather remarkable, especially in view of the substantial savings caused by the approximations: on a Sun Ultra II, the computational cost of learning was 138 min/iteration in the exact case, vs. 6 min/iteration for the 5+5 clustering, and less than 5 min/iteration for the other two.

## 4   Online learning

Our analysis also gives us the tools to address another important problem with learning dynamic models: the need to reason about the entire temporal sequence at once. One consequence of our contraction result is that the effect of approximations done far away in the sequence decays exponentially with the time difference. In particular, the effect of an approximation which ignores observations that are far in the future is also limited. Therefore, if we do inference for a time slice based on a small window of observations into the future, the result should still be fairly accurate. More precisely, assume that we are at time $t$ and are considering a window of size $w$. We can view the uniform message as a very bad approximation to $\beta^{(t+w)}$. But as we propagate this approximate backward message from $t + w$ to $t$, the error will decay exponentially with $w$.

Based on these insights, we experimented with various online algorithms that use a small window approximation. Our online algorithms are based on the approach of [11], in which ESS are updated with an exponential decay every few data cases; the parameters are then updated correspondingly. The main problem with frequent parameter updates in the online setting is that they require a recomputation of the messages computed using the old parameters. For long sequences, the computational cost of such a scheme would be prohibitive. In our algorithms, we simply leave the forward messages unchanged, under the assumption that the most recent time slices used parameters that are very close to the new ones. Our contraction result tells us that the use of old parameters far back in the sequence has a negligible effect on the message. We tried several schemes for the update of the backward messages. In the *dynamic-1000* approach, we use a backward message computed over 1000 slices, with the closer messages recomputed very frequently as the parameters are changed, based on cached messages that used older parameters. The 8

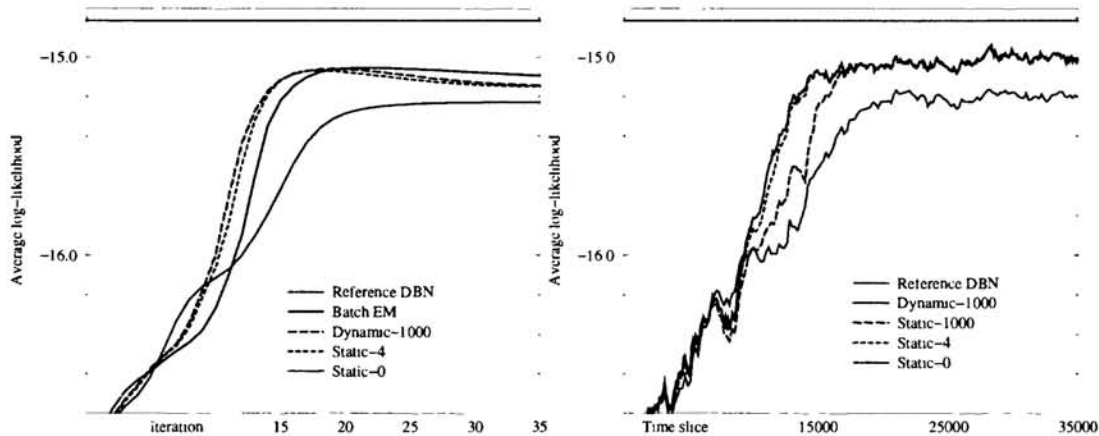

Figure 2: Temporal approximations for (a) batch setting; (b) online setting.

closest messages are updated every parameter update, the next 16 every other update, etc. This approach is the closest realistic alternative to a full update of backward messages. In the *static-1000* approach, we use a very long window (1000 slices), but do not recompute messages; when the window ends, we use the current parameters to compute the messages for the entire next window. In the *static-4* approach, we do the same, but use a very short window of 4 slices. Finally, in the *static-0* approach, there is no lookahead at all; only the past and present evidence is used to compute the joint beliefs. The latter case is often used (e.g., in the context of Kalman filters [9]) for online learning of the process parameters. To minimize the computational burden, all tests were conducted using the 5+5 structural approximation. The running time for the various algorithms are: 0.4 sec/slice for batch EM; 1.4 for dynamic-1000; 0.5 for static-1000 and for static-4; and 0.3 for static-0.

We evaluated these temporal approximations both in an online and in a batch setting. In the batch experiments, we used the same 1000-step sequence used above. The results are shown in Figure 2(a). We see that the dynamic-1000 algorithm reaches the same quality model as standard batch EM, but converges sooner. As in [11], the difference is due to the frequent update of the sufficient statistics based on more accurate parameters. More interestingly, we see that the static-4 algorithm, which uses a lookahead of only 4, also reaches the same accuracy. Thus, our approximation—ignoring evidence far in the future—is a good one, even for a very weak notion of "far". By contrast, we see that the quality reached by the static-0 approach is significantly lower: the sufficient statistics used by the EM algorithm in this case are consistently worse, as they ignore all future evidence. Thus, in this network, a window of size 4 is as good as full forward-backward, whereas one of size 0 is clearly worse. Our online learning experiments, shown in Figure 2(b), used a single long sequence of 40,000 slices. Again, we see that the static-4 approach is almost indistinguishable in terms of accuracy from the dynamic-1000 approach, and that both converge more rapidly than the static-1000 algorithm. Thus, frequent updates over short windows are better than infrequent updates over longer ones. Finally, we see again that the static-0 algorithm converges to a hypothesis of much lower quality. Thus, even a very short window allows rapid convergence to the "best possible" answer, but a window of size 0 does not.

## 5   Conclusion and extensions

In this paper, we suggested the use of simple structural approximations in the inference algorithm used in an E-step. Our results suggest that even severe structural approximations have almost negligible effects on the accuracy of learning. The advantages of approximate inference in the learning setting are even more pronounced than in the inference task [2], as the small errors caused by approximation are negligible compared to the larger ones

induced by the learning process. Our techniques provide a new and simple approach for learning structured models of complex dynamic systems, with the resulting advantages of generalization and the ability to incorporate prior knowledge. We also presented a new algorithm for the online learning task, showing that we can learn high-quality models using a very small time window of future observations.

The work most comparable to ours is the variational approach to approximate inference applied to learning factorial HMMs [8]. While we have not done a direct empirical comparison, it seems likely that the variational approach would work better for densely connected models, whereas our approach would dominate for structured models such as the one in our experiments. Indeed, for this model, our algorithms track exact EM so closely that any significant improvement in accuracy is unlikely. Our algorithm is also simpler and easier to implement. Most importantly, it is applicable to the task of online learning.

The most obvious extension to our results is an integration of our ideas with structure learning algorithm for DBNs [7]. We believe that the resulting algorithm will be able to learn structured models for real-life complex systems.

**Acknowledgements.** We thank Tim Huang for providing us with the BAT network, and Nir Friedman and Leonid Gurvits for useful discussions. This research was supported by ARO under the MURI program "Integrated Approach to Intelligent Systems", and by DARPA contract DACA76-93-C-0025 under subcontract to IET, Inc.

# References

[1] M. Artzrouni and X. Li. A note on the coefficient of ergodicity of a column-allowable nonnegative matrix. *Linear algebra and applications*, 214:93–101, 1995.

[2] X. Boyen and D. Koller. Tractable inference for complex stochastic processes. In *Proc. UAI*, pages 33–42, 1998.

[3] T. Cover and J. Thomas. *Elements of Information Theory*. Wiley, 1991.

[4] T. Dean and K. Kanazawa. A model for reasoning about persistence and causation. *Comp. Int.*, 5(3), 1989.

[5] A.P. Dempster, N.M. Laird, and D.B. Rubin. Maximum-likelihood from incomplete data via the EM algorithm. *Journal of the Royal Statistical Society*, B39:1–38, 1977.

[6] J. Forbes, T. Huang, K. Kanazawa, and S.J. Russell. The BATmobile: Towards a Bayesian automated taxi. In *Proc. IJCAI*, 1995.

[7] N. Friedman, K. Murphy, and S.J. Russell. Learning the structure of dynamic probabilistic networks. In *Proc. UAI*, pages 139–147, 1998.

[8] Z. Ghahramani and M.I. Jordan. Factorial hidden Markov models. In *NIPS 8*, 1996.

[9] R.E. Kalman. A new approach to linear filtering and prediction problems. *J. of Basic Engineering*, 82:34–45, 1960.

[10] S.L. Lauritzen and D.J. Spiegelhalter. Local computations with probabilities on graphical structures and their application to expert systems. *J. Roy. Stat. Soc.*, B 50, 1988.

[11] R.M. Neal and G.E. Hinton. A view of the EM algorithm that justifies incremental, sparse, and other variants. In M.I. Jordan, editor, *Learning in Graphical Models*. Kluwer, 1998.

[12] L. Rabiner and B. Juang. An introduction to hidden Markov models. *IEEE Acoustics, Speech & Signal Processing*, 1986.

[13] G. Zweig and S.J. Russell. Speech recognition with dynamic bayesian networks. In *Proc. AAAI*, pages 173–180, 1998.
